# Unsupervised Variational Bayesian Learning of Nonlinear Models

**Antti Honkela** and **Harri Valpola**
Neural Networks Research Centre, Helsinki University of Technology
P.O. Box 5400, FI-02015 HUT, Finland
{Antti.Honkela, Harri.Valpola}@hut.fi
http://www.cis.hut.fi/projects/bayes/

## Abstract

In this paper we present a framework for using multi-layer perceptron (MLP) networks in nonlinear generative models trained by variational Bayesian learning. The nonlinearity is handled by linearizing it using a Gauss–Hermite quadrature at the hidden neurons. This yields an accurate approximation for cases of large posterior variance. The method can be used to derive nonlinear counterparts for linear algorithms such as factor analysis, independent component/factor analysis and state-space models. This is demonstrated with a nonlinear factor analysis experiment in which even 20 sources can be estimated from a real world speech data set.

## 1 Introduction

Linear latent variable models such as factor analysis, principal component analysis (PCA) and independent component analysis (ICA) [1] are used in many applications ranging from engineering to social sciences and psychology. In many of these cases, the effect of the desired factors or sources to the observed data is, however, not linear. A nonlinear model could therefore produce better results.

The method presented in this paper can be used as a basis for many nonlinear latent variable models, such as nonlinear generalizations of the above models. It is based on the variational Bayesian framework, which provides a solid foundation for nonlinear modeling that would otherwise be prone to overfitting [2]. It also allows for easy comparison of different model structures, which is even more important for flexible nonlinear models than for simpler linear models.

General nonlinear generative models for data $\mathbf{x}(t)$ of the type

$$\mathbf{x}(t) = \mathbf{f}(\mathbf{s}(t), \boldsymbol{\theta_f}) + \mathbf{n}(t) = \mathbf{B}\boldsymbol{\phi}(\mathbf{A}\mathbf{s}(t) + \mathbf{a}) + \mathbf{b} + \mathbf{n}(t) \tag{1}$$

often employ a multi-layer perceptron (MLP) (as in the equation) or a radial basis function (RBF) network to model the nonlinearity. Here $\mathbf{s}(t)$ are the latent variables of the model, $\mathbf{n}(t)$ is noise and $\boldsymbol{\theta_f}$ are the parameters of the nonlinearity, in case of MLP the weight matrices $\mathbf{A}, \mathbf{B}$ and bias vectors $\mathbf{a}, \mathbf{b}$. In context of variational Bayesian methods, RBF networks seem more popular of the two because it is easier

to evaluate analytic expressions and bounds for certain key quantities [3]. With MLP networks such values are not as easily available and one usually has to resort to numeric approximations. Nevertheless, MLP networks can often, especially for nearly linear models and in high dimensional spaces, provide an equally good model with fewer parameters [4]. This is important with generative models whose latent variables are independent or at least uncorrelated and the intrinsic dimensionality of the input is large. A reasonable approximate bound for a good model is also often better than a strict bound for a bad model.

Most existing applications of variational Bayesian methods for nonlinear models are concerned with the supervised case where the inputs of the network are known and only the weights have to be learned [3, 5]. This is easier as there are fewer parameters with related posterior variance above the nonlinear hidden layer and the distributions thus tend to be easier to handle.

In this paper we present a novel method for evaluating the statistics of the outputs of an MLP network in context of unsupervised variational Bayesian learning of its weights and inputs. The method is demonstrated with a nonlinear factor analysis problem. The new method allows for reliable estimation of a larger number of factors than before [6, 7].

## 2   Variational learning of unsupervised MLPs

Let us denote the observed data by $\boldsymbol{X} = \{\mathbf{x}(t)|t\}$, the latent variables of the model by $\boldsymbol{S} = \{\mathbf{s}(t)|t\}$ and the model parameters by $\boldsymbol{\theta} = (\theta_i)$. The nonlinearity (1) can be used as a building block of many different models depending on the model assumed for the sources $\boldsymbol{S}$. Simple Gaussian prior on $\boldsymbol{S}$ leads to a nonlinear factor analysis (NFA) model [6, 7] that is studied here because of its simplicity. The method could easily be extended with a mixture-of-Gaussians prior on $\boldsymbol{S}$ [8] to get a nonlinear independent factor analysis model, but this is omitted here. In many nonlinear blind source separation (BSS) problems it is enough to apply simple NFA followed by linear ICA postprocessing to achieve nonlinear BSS [6, 7]. Another possible extension would be to include dynamics for $\boldsymbol{S}$ as in [9].

In order to deal with the flexible nonlinear models, a powerful learning paradigm resistant to overfitting is needed. The variational Bayesian method of ensemble learning [2] has proven useful here. Ensemble learning is based on approximating the true posterior $p(\boldsymbol{S}, \boldsymbol{\theta}|\boldsymbol{X})$ with a tractable approximation $q(\boldsymbol{S}, \boldsymbol{\theta})$, typically a multivariate Gaussian with a diagonal covariance. The approximation is fitted to minimize the cost

$$\mathcal{C} = \left\langle \log \frac{q(\boldsymbol{S}, \boldsymbol{\theta})}{p(\boldsymbol{S}, \boldsymbol{\theta}, \boldsymbol{X})} \right\rangle = D(q(\boldsymbol{S}, \boldsymbol{\theta}) || p(\boldsymbol{S}, \boldsymbol{\theta}|\boldsymbol{X})) - \log p(\boldsymbol{X}) \qquad (2)$$

where $\langle \cdot \rangle$ denotes expectation over $q(\boldsymbol{S}, \boldsymbol{\theta})$ and $D(q||p)$ is the Kullback-Leibler divergence between $q$ and $p$. As the Kullback-Leibler divergence is always non-negative, $\mathcal{C}$ yields an upper bound for $-\log p(\boldsymbol{X})$ and thus a lower bound for the evidence $p(\boldsymbol{X})$. The cost can be evaluated analytically for a large class of mainly linear models [10, 11] leading to simple and efficient learning algorithms.

### 2.1   Evaluating the cost

Unfortunately, the cost (2) cannot be evaluated analytically for the nonlinear model (1). Assuming a Gaussian noise model, the likelihood term of $\mathcal{C}$ becomes

$$\mathcal{C}_x = \langle -\log p(\boldsymbol{X}|\boldsymbol{S}, \boldsymbol{\theta}) \rangle = \sum_t \langle -\log N(\mathbf{x}(t); \ \mathbf{f}(\mathbf{s}(t), \boldsymbol{\theta_f}), \boldsymbol{\Sigma_x}) \rangle . \qquad (3)$$

The term $\mathcal{C}_x$ depends on the first and second moments of $\mathbf{f}(\mathbf{s}(t), \boldsymbol{\theta_f})$ over the posterior approximation $q(\boldsymbol{S}, \boldsymbol{\theta})$, and they cannot easily be evaluated analytically. Assuming the noise covariance is diagonal, the cross terms of the covariance of the output are not needed, only the scalar variances of the different components.

If the activation functions of the MLP network were linear, the output mean and variance could be evaluated exactly using only the mean and variance of the inputs $\mathbf{s}(t)$ and $\boldsymbol{\theta_f}$. Thus a natural first approximation would be to linearize the network about the input mean using derivatives [6]. Taking the derivative with respect to $\mathbf{s}(t)$, for instance, yields

$$\frac{\partial \mathbf{f}(\mathbf{s}(t), \boldsymbol{\theta_f})}{\partial \mathbf{s}(t)} = \mathbf{B} \, \mathrm{diag}(\boldsymbol{\phi}'(\mathbf{y}(t))) \, \mathbf{A}, \tag{4}$$

where $\mathrm{diag}(\mathbf{v})$ denotes a diagonal matrix with elements of vector $\mathbf{v}$ on the main diagonal and $\mathbf{y}(t) = \mathbf{A}\mathbf{s}(t) + \mathbf{a}$. Due to the local nature of the approximation, this can lead to severe underestimation of the variance, especially when the hidden neurons of the MLP network operate in the saturated region. This makes the nonlinear factor analysis algorithm using this approach unstable with large number of factors because the posterior variance corresponding to the last factors is typically large.

To avoid this problem, we propose using a Gauss–Hermite quadrature to evaluate an effective linearization of the nonlinear activation functions $\phi(y_i(t))$. The Gauss–Hermite quadrature is a method for approximating weighted integrals

$$\int_{-\infty}^{\infty} f(x) \, \exp(-x^2) \, dx \approx \sum_k w_k \, f(t_k), \tag{5}$$

where the weights $w_k$ and abscissas $t_k$ are selected by requiring exact result for suitable number of low-order polynomials. This allows evaluating the mean and variance of $\phi(y_i(t))$ by quadratures

$$\overline{\phi}(y_i(t))_{\mathrm{GH}} = \sum_k w'_k \phi\left(\overline{y}_i(t) + t'_k \sqrt{\widetilde{y}_i(t)}\right) \tag{6}$$

$$\widetilde{\phi}(y_i(t))_{\mathrm{GH}} = \sum_k w'_k \left[\phi\left(\overline{y}_i(t) + t'_k \sqrt{\widetilde{y}_i(t)}\right) - \overline{\phi}(y_i(t))_{\mathrm{GH}}\right]^2, \tag{7}$$

respectively. Here the weights and abscissas have been scaled to take into account the Gaussian pdf weight instead of $\exp(-x^2)$, and $\overline{y}_i(t)$ and $\widetilde{y}_i(t)$ are the mean and variance of $y_i(t)$, respectively. We used a three point quadrature that yields accurate enough results but can be evaluated quickly. Using e.g. five points improves the accuracy slightly, but slows the computation down significantly. As both of the quadratures depend on $\phi$ at the same points, they can be evaluated together easily.

Using the approximation formula $\widetilde{\phi}(y_i(t)) = \phi'(y_i(t))^2 \widetilde{y}_i(t)$, the resulting mean and variance can be interpreted to yield an effective linearization of $\phi(y_i(t))$ through

$$\langle\phi(y_i(t))\rangle := \overline{\phi}(y_i(t))_{\mathrm{GH}} \qquad \langle\phi'(y_i(t))\rangle := \sqrt{\frac{\widetilde{\phi}(y_i(t))_{\mathrm{GH}}}{\widetilde{y}_i(t)}}. \tag{8}$$

The positive square root is used here because the derivative of the logistic sigmoid used as activation function is always positive. Using these to linearize the MLP as in Eq. (4), the exact mean and variance of the linearized model can be evaluated in a relatively straightforward manner. Evaluation of the variance due to the sources requires propagating matrices through the network to track the correlations between the hidden units. Hence the computational complexity depends quadratically on the number of sources. The same problem does not affect the network weights as each parameter only affects the value of one hidden neuron.

## 2.2 Details of the approximation

The mean and variance of $\phi(y_i(t))$ depend on the distribution of $y_i(t)$. The Gauss–Hermite quadrature assumes that $y_i(t)$ is Gaussian. This is not true in our case, as the product of two independent normally distributed variables $a_{ij}$ and $s_j(t)$ is super-Gaussian, although rather close to Gaussian if the mean of one of the variables is significantly larger in absolute value than the standard deviation. In case of $N$ sources, the actual input $y_i(t)$ is a sum of $N$ of these and a Gaussian variable and therefore rather close to a Gaussian, at least for larger values of $N$.

Ignoring the non-Gaussianity, the quadrature depends on the mean and variance of $y_i(t)$. These can be evaluated exactly because of the linearity of the mapping as

$$\widetilde{y}_{i,\text{tot}}(t) = \sum_j \left( \widetilde{A}_{ij}(\overline{s}_j(t)^2 + \widetilde{s}_j(t)) + \overline{A}_{ij}^2 \widetilde{s}_j(t) \right) + \widetilde{a}_i, \tag{9}$$

where $\overline{\theta}$ denotes the mean and $\widetilde{\theta}$ the variance of $\theta$. Here it is assumed that the posterior approximations $q(\boldsymbol{S})$ and $q(\boldsymbol{\theta_f})$ have diagonal covariances. Full covariances can be used instead without too much difficulty, if necessary.

In an experiment investigating the approximation accuracy with a random MLP [12], the Taylor approximation was found to underestimate the output variance by a factor of 400, at worst. The worst case result of the above approximation was underestimation by a factor of 40, which is a great improvement over the Taylor approximation, but still far from perfect. The worst case behavior could be improved to underestimation by a factor of 5 by introducing another quadrature evaluated with a different variance for $y_i(t)$. This change cannot be easily justified except by the fact that it produces better results. The difference in behavior of the two methods in more realistic cases is less drastic, but the version with two quadratures seems to provide more accurate approximations.

The more accurate approximation is implemented by evaluating another quadrature using the variance of $y_i(t)$ originating mainly from $\boldsymbol{\theta_f}$,

$$\widetilde{y}_{i,\text{weight}}(t) = \sum_j \widetilde{A}_{ij}(\overline{s}_j(t)^2 + \widetilde{s}_j(t)) + \widetilde{a}_i, \tag{10}$$

and using the implied $\langle \phi'(y_i(t)) \rangle$ in the evaluation of the effects of these variances. The total variance (9) is still used in evaluation of the means and the evaluation of the effects of the variance of $\mathbf{s}(t)$.

## 2.3 Learning algorithm for nonlinear factor analysis

The nonlinear factor analysis (NFA) model [6] is learned by numerically minimizing the cost $\mathcal{C}$ evaluated above. The minimization algorithm is a combination of conjugate gradient for the means of $\boldsymbol{S}$ and $\boldsymbol{\theta_f}$, fixed point iteration for the variances of $\boldsymbol{S}$ and $\boldsymbol{\theta_f}$, and EM like updates for other parameters and hyperparameters.

The fixed point update algorithm for the variances follows from writing the cost function as a sum

$$\mathcal{C} = \mathcal{C}_q + \mathcal{C}_p = \langle \log q(\boldsymbol{S}, \boldsymbol{\theta}) \rangle + \langle -\log p(\boldsymbol{S}, \boldsymbol{\theta}, \boldsymbol{X}) \rangle. \tag{11}$$

A parameter $\theta_i$ that is assumed independent of others under $q$ and has a Gaussian posterior approximation $q(\theta_i) = N(\theta_i; \overline{\theta}_i, \widetilde{\theta}_i)$, only affects the corresponding negentropy term $-1/2 \log(2\pi e \widetilde{\theta}_i)$ in $\mathcal{C}_q$. Differentiating this with respect to $\widetilde{\theta}_i$ and setting the result to zero leads to a fixed point update rule $\widetilde{\theta}_i = \left( 2\partial \mathcal{C}_p / \partial \widetilde{\theta}_i \right)^{-1}$. In order to

get a stable update algorithm for the variances, dampening by halving the step on log scale until the cost function does not increase must be added to the fixed point updates. The variance is increased at most by 10 % on one iteration and not set to a negative value even if the gradient is negative.

The required partial derivatives can be evaluated analytically with simple back-propagation like computations with the MLP network. The quadratures used at hidden nodes lead to analytical expressions for the means and variances of the hidden nodes and the corresponding feedback gradients are easy to derive. Along with the derivatives with respect to variances, it is easy to evaluate them with respect to means of the same parameters. These derivatives can then be used in a conjugate gradient algorithm to update the means of $S$ and $\boldsymbol{\theta_f}$.

Due to the flexibility of the MLP network and the gradient based learning algorithm, the nonlinear factor analysis method is sensitive to the initialization. We have used linear PCA for initialization of the means of the sources $S$. The means of the weights $\boldsymbol{\theta_f}$ are initialized randomly while all the variances are initialized to small constant values. After this, the sources are kept fixed for 20 iterations while only the network weights are updated. The hyperparameters governing noise and parameter distributions are only updated after 80 more iterations to update the sources and the MLP. By that time, a reasonable model of the data has been learned and the method is not likely to prune away all the sources and other parameters as unnecessary.

## 2.4   Other approximation methods

Another way to get a more robust approximation for the statistics of $\mathbf{f}$ would be to use the deterministic sampling approach used in unscented transform [13] and consecutively in different unscented algorithms. Unfortunately this approach does not work very well in high dimensional cases. The unscented transform also ignores all the prior information on the form of the nonlinearity. In case of the MLP network, everything except the scalar activation functions is known to be linear. All information on the correlations of variables is also ignored, which leads to loss of accuracy when the output depends on products of input variables like in our case. In an experiment of mean and log-variance approximation accuracy with a relatively large random MLP [12], the unscented transform needed over 100 % more time to achieve results with 10 times the mean squared error of the proposed approach.

Part of our problem was also faced by Barber and Bishop in their work on ensemble learning for supervised learning of MLP networks [5]. In their work the inputs $\mathbf{s}(t)$ of the network are part of the data and thus have no associated variance. This makes the problem easier as the inputs $\mathbf{y}(t)$ of the hidden neurons are Gaussian. By using the cumulative Gaussian distribution or the error function erf as the activation function, the mean of the outputs of the hidden neurons and thus of the outputs of the whole network can be evaluated analytically. The covariances still need to be evaluated numerically, and that is done by evaluating all the correlations of the hidden neurons separately. In a network with $H$ hidden neurons, this requires $\mathcal{O}(H^2)$ quadrature evaluations.

In our case the inputs of the hidden neurons are not Gaussian and hence even the error function as the activation function would not allow for exact evaluation of the means. This is why we have decided to use the standard logistic sigmoid activation function in form of tanh which is more common and faster to evaluate numerically. In our approach all the required means and variances can be evaluated with $\mathcal{O}(H)$ quadratures.

# 3 Experiments

The proposed nonlinear factor analysis method was tested on natural speech data set consisting of spectrograms of 24 individual words of Finnish speech, spoken by 20 different speakers. The spectra were modified to mimic the reception abilities of the human ear. This is a standard preprocessing procedure for speech recognition. No speaker or word information was used in learning, the spectrograms of different words were simply blindly concatenated. The preprocessed data consisted of 2547 30-dimensional spectrogram vectors.

The data set was tested with two different learning algorithms for the NFA model, one based on the Taylor approximation introduced in [6] and another based on the proposed approximation. Contrary to [6], the algorithm based on Taylor approximation used the same conjugate gradient based optimization algorithm as the new approximation. This helped greatly in stabilizing the algorithm that used to be rather unstable with high source dimensionalities due to sensitivity of the Taylor approximation in regions where it is not really valid. Both algorithms were tested using 1 to 20 sources, each number with four different random initializations for the MLP network weights. The number of hidden neurons in the MLP network was 40. The learning algorithm was run for 2000 iterations.[1]

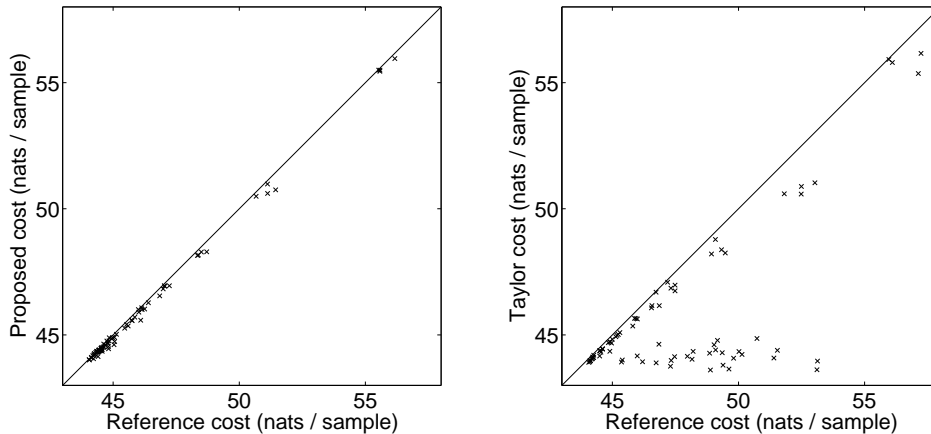

Figure 1: The attained values of $\mathcal{C}$ in different simulations as evaluated by the different approximations plotted against reference values evaluated by sampling. The left subfigure shows the values from experiments using the proposed approximation and the right subfigure from experiments using the Taylor approximation.

Fig. 1 shows a comparison of the cost function values evaluated by the different approximations and a reference value evaluated by sampling. The reference cost values were evaluated by sampling 400 points from the distribution $q(\boldsymbol{S}, \boldsymbol{\theta_f})$, evaluating $\mathbf{f}(\mathbf{s}, \boldsymbol{\theta_f})$ at those points, and using the mean and variance of the output points in the cost function evaluation. The accuracy of the procedure was checked by performing the evaluation 100 times for one of the simulations. The standard deviation of the values was $5 \cdot 10^{-3}$ nats per sample which should not show at all in the figures. The unit *nat* here signifies the use of natural logarithm in Eq. (2).

The results in Fig. 1 show that the proposed approximation yields consistently very

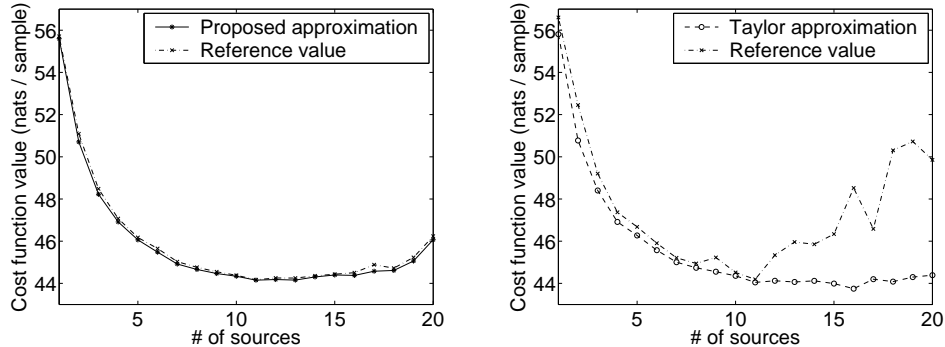

Figure 2: The attained value of $\mathcal{C}$ in simulations with different numbers of sources. The values shown are the means of 4 simulations with different random initializations. The left subfigure shows the values from experiments using the proposed approximation and the right subfigure from experiments using the Taylor approximation. Both values are compared to reference values evaluated by sampling.

reliable estimates of the true cost, although it has a slight tendency to underestimate it. The older Taylor approximation [6] breaks down completely in some cases and reports very small costs even though the true value can be significantly larger.

The situations where the Taylor approximation fails are illustrated in Fig. 2, which shows the attained cost as a function of number of sources used. The Taylor approximation shows a decrease in cost as the number of the sources increases even though the true cost is increasing rapidly. The behavior of the proposed approximation is much more consistent and qualitatively correct.

## 4 Discussion

The problem of estimating the statistics of a nonlinear transform of a probability distribution is also encountered in nonlinear extensions of Kalman filtering. The Taylor approximation corresponds to extended Kalman filter and the new approximation can be seen as a modification of it with a more accurate linearization. This opens up many new potential applications in time series analysis and elsewhere. The proposed method is somewhat similar to unscented Kalman filtering based on the unscented transform [13], but much better suited for high dimensional MLP-like nonlinearities. This is not very surprising, as worst case complexity of general Gaussian integration is exponential with respect to the dimensionality of the input [14] and unscented transform as a general method with linear complexity is bound to be less accurate in high dimensional problems. In case of the MLP, the complexity of the unscented transform depends on the number of all weights, which in our case with 20 sources can be more than 2000.

## 5 Conclusions

In this paper we have proposed a novel approximation method for unsupervised MLP networks in variational Bayesian learning. The approximation is based on using numerical Gauss–Hermite quadratures to evaluate the global effect of the nonlinear activation function of the network to produce an effective linearization of the MLP. The statistics of the outputs of the linearized network can be evaluated

exactly to get accurate and reliable estimates of the statistics of the MLP outputs. These can be used to evaluate the standard variational Bayesian ensemble learning cost function $\mathcal{C}$ and numerically minimize it using a hybrid fixed point / conjugate gradient algorithm.

We have demonstrated the method with a nonlinear factor analysis model and a real world speech data set. It was able to reliably estimate all the 20 factors we attempted from the 30-dimensional data set. The presented method can be used together with linear ICA for nonlinear BSS [7], and the approximation can be easily applied to more complex models such as nonlinear independent factor analysis [6] and nonlinear state-space models [9].

### Acknowledgments

The authors wish to thank David Barber, Markus Harva, Bert Kappen, Juha Karhunen, Uri Lerner and Tapani Raiko for useful comments and discussions. This work was supported in part by the IST Programme of the European Community, under the PASCAL Network of Excellence, IST-2002-506778. This publication only reflects the authors' views.

## Footnotes

[1]The Matlab code used in the experiments is available at `http://www.cis.hut.fi/projects/bayes/software/`.

## References

[1] A. Hyvärinen, J. Karhunen, and E. Oja. *Independent Component Analysis*. J. Wiley, 2001.

[2] G. E. Hinton and D. van Camp. Keeping neural networks simple by minimizing the description length of the weights. In *Proc. of the 6th Ann. ACM Conf. on Computational Learning Theory*, pp. 5–13, Santa Cruz, CA, USA, 1993.

[3] P. Sykacek and S. Roberts. Adaptive classification by variational Kalman filtering. In *Advances in Neural Information Processing Systems 15*, pp. 753–760. MIT Press, 2003.

[4] S. Haykin. *Neural Networks – A Comprehensive Foundation, 2nd ed.* Prentice-Hall, 1999.

[5] D. Barber and C. Bishop. Ensemble learning for multi-layer networks. In *Advances in Neural Information Processing Systems 10*, pp. 395–401. MIT Press, 1998.

[6] H. Lappalainen and A. Honkela. Bayesian nonlinear independent component analysis by multi-layer perceptrons. In M. Girolami, ed., *Advances in Independent Component Analysis*, pp. 93–121. Springer-Verlag, Berlin, 2000.

[7] H. Valpola, E. Oja, A. Ilin, A. Honkela, and J. Karhunen. Nonlinear blind source separation by variational Bayesian learning. *IEICE Transactions on Fundamentals of Electronics, Communications and Computer Sciences*, E86-A(3):532–541, 2003.

[8] H. Attias. Independent factor analysis. *Neural Computation*, 11(4):803–851, 1999.

[9] H. Valpola and J. Karhunen. An unsupervised ensemble learning method for nonlinear dynamic state-space models. *Neural Computation*, 14(11):2647–2692, 2002.

[10] H. Attias. A variational Bayesian framework for graphical models. In *Advances in Neural Information Processing Systems 12*, pp. 209–215. MIT Press, 2000.

[11] Z. Ghahramani and M. Beal. Propagation algorithms for variational Bayesian learning. In *Advances in Neural Information Processing Systems 13*, pp. 507–513. MIT Press, 2001.

[12] A. Honkela. Approximating nonlinear transformations of probability distributions for nonlinear independent component analysis. In *Proc. 2004 IEEE Int. Joint Conf. on Neural Networks (IJCNN 2004)*, pp. 2169–2174, Budapest, Hungary, 2004.

[13] S. Julier and J. K. Uhlmann. A general method for approximating nonlinear transformations of probability distributions. Technical report, Robotics Research Group, Department of Engineering Science, University of Oxford, 1996.

[14] F. Curbera. Delayed curse of dimension for Gaussian integration. *Journal of Complexity*, 16(2):474–506, 2000.
